# Learning Macro-Actions in Reinforcement Learning

**Jette Randløv**
Niels Bohr Inst., Blegdamsvej 17,
University of Copenhagen,
DK-2100 Copenhagen Ø, Denmark
randlov@nbi.dk

## Abstract

We present a method for automatically constructing macro-actions from scratch from primitive actions during the reinforcement learning process. The overall idea is to reinforce the tendency to perform action $b$ after action $a$ if such a pattern of actions has been rewarded. We test the method on a bicycle task, the car-on-the-hill task, the race-track task and some grid-world tasks. For the bicycle and race-track tasks the use of macro-actions approximately halves the learning time, while for one of the grid-world tasks the learning time is reduced by a factor of 5. The method did not work for the car-on-the-hill task for reasons we discuss in the conclusion.

## 1 INTRODUCTION

A macro-action is a sequence of actions chosen from the primitive actions of the problem.[1] Lumping actions together as macros can be of great help for solving large problems (Korf, 1985a,b; Gullapalli, 1992) and can sometimes greatly speed up learning (Iba, 1989; McGovern, Sutton & Fagg, 1997; McGovern & Sutton, 1998; Sutton, Precup & Singh, 1998; Sutton, Singh, Precup & Ravindran, 1999). Macro-actions might be essential for scaling up reinforcement learning to very large problems. Construction of macro-actions by hand requires insight into the problem at hand. It would be more elegant and useful if the agent itself could decide what actions to lump together (Iba, 1989; McGovern & Sutton, 1998; Sutton, Precup & Singh, 1998; Hauskrecht et al., 1998).

## 2  ACTION-TO-ACTION MAPPING

In reinforcement learning we want to learn a mapping from states to actions, $s \to a$ that maximizes the total expected reward (Sutton & Barto, 1998). Sometimes it might be of use to learn a mapping from actions to actions as well. We believe that acting according to an action-to-action mapping can be useful for three reasons:

1. During the early stages of learning the agent will enter areas of the state space it has never visited before. If the agent acts according to an action-to-action mapping it might be guided through such areas where there is yet no clear choice of action otherwise. In other words it is much more likely that an action-to-action mapping could guide the agent to perform almost optimally in states never visited than a random policy.

2. In some situations, for instance in an emergency, it can be useful to perform a certain open-loop sequence of actions, without being guided by state information. Consider for instance an agent learning to balance on a bicycle (Randløv & Alstrøm, 1998). If the bicycle is in an unbalanced state, the agent must forget about the position of the bicycle and carry out a sequence of actions to balance the bicycle again. Some of the state information—the position of the bicycle relative to some goal—does not matter, and might actually distract the agent, while the history of the most recent actions might contain just the needed information to pick the next action.

3. An action-to-action mapping might lead the agent to explore the relevant areas of the state space in an efficient way instead of just hitting them by chance.

We therefore expect that learning an action-to-action mapping in addition to a state-action mapping can lead to faster overall learning. Even though the system has the Markov property, it may be useful to remember a bit of the action history. We want the agent to perform a sequence of actions while being aware of the development of the states, but not only being controlled by the states.

Many people have tried to deal with imperfect state information by adding memory of previous states and actions to the information the agent receives (Andreae & Cashin, 1969; McCallum, 1995; Hansen, Barto & Zilberstein, 1997; Burgard et al., 1998). In this work we are not specially concerned with non-Markov problems. However the results in this paper suggest that some methods for partially observable MDP could be applied to MDPs and result in faster learning.

The difficult part is how to combine the suggestion made by the action-to-action mapping with the conventional state-to-action mapping. Obviously we do not want to learn the mapping $(s_t, a_{t-1}) \to a_t$ on tabular form, since that would destroy the possibility of using the action-to-action mapping generalisation over the state space.

In our approach we decided to learn two value mappings. The mapping $Q_s$ is the conventional $Q$-value normally used for state-to-action mapping, while the mapping $Q_a$ represents the value belonging to the action-to-action mapping. When making a choice, we add the $Q$-values of the suggestions made by the two mappings, normalize and use the new values to pick an action in the usual way:

$$\bar{Q}(s_t, a_{t-1}, a_t) = \frac{Q_s(s_t, a_t) + \beta\, Q_a(a_{t-1}, a_t)}{1 + \beta}.$$

Here $\bar{Q}$ is the $Q$-value that we actually use to pick the next action. The parameter $\beta$ determines the influence of the action-to-action mapping. For $\beta = 0$ we are back with the usual $Q$-values. The idea is to reinforce the tendency to perform action $b$ after action $a$ if such a pattern of actions is rewarded. In this way the agent forms habits or macro-actions and it will sometimes act according to them.

# 3  RESULTS

How do we implement an action-to-action mapping and the $\bar{Q}$-values? Many algorithms have been developed to find near optimal state-to-action mappings on a trial-and-error basis. An example of such a algorithm is Sarsa($\lambda$), developed by Rummery and Niranjan (Rummery & Niranjan, 1994; Rummery, 1995). We use Sarsa($\lambda$) with replacing eligibility traces (Singh & Sutton, 1996) and table look-up. Eligibility traces are attached to the $Q_a$-values—one for each action-action pair.[2] During learning the $Q_s$ and $Q_a$-values are both adjusted according to the overall TD error $\delta_t = r_{t+1} + \gamma \bar{Q}_t(s_{t+1}, a_{t+1}) - \bar{Q}_t(s_t, a_t)$. The update for the $Q_a$-values has the form $\Delta Q_a(a_{t-1}, a_t) = \beta \delta e(a_{t-1}, a_t)$. For description of

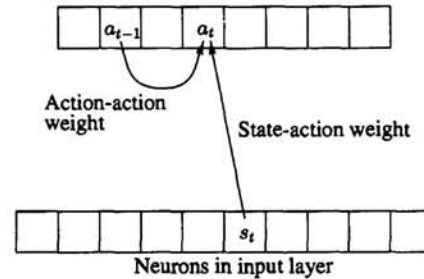

Figure 1: One can think of the action-to-action mapping in terms of weights between output neurons in a network calculating the $Q$-value.

the Sarsa($\lambda$)-algorithm see Rummery (1995) or Sutton & Barto (1998). Figure 1 shows the idea in terms of a neural network with no hidden layers. The new $Q_a$-values correspond to weights from output neurons to output neurons.

## 3.1  THE BICYCLE

We first tested the new $\bar{Q}$-values on a bicycle system. To solve this problem the agent has to learn to balance a bicycle for 1000 seconds and thereby ride 2.8 km. At each time step the agent receives information about the state of the bicycle: the angle and angular velocity of the handlebars, the angle, angular velocity and angular acceleration of the angle of the bicycle from vertical.

The agent chooses two basic actions: the torque that should be applied to the handle bars, and how much the centre of mass should be displaced from the bicycle's plan—a total of 9 possible actions (Randløv & Alstrøm, 1998). The reward at each time step is 0 unless the bicycle has fallen, in which case it is $-1$. The agent uses $\alpha = 0.5$, $\gamma = 0.99$ and $\lambda = 0.95$. For further description and the equations for the system we refer the reader to the original paper. Figure 2 shows how the learning time varies with the value of $\beta$. The error bars show the standard error in all graphs. For small values of $\beta$ ($\approx 0.03$) the agent learns the task faster than with usual Sarsa($\lambda$) ($\beta = 0$). As expected, large values of $\beta$ slow down learning.

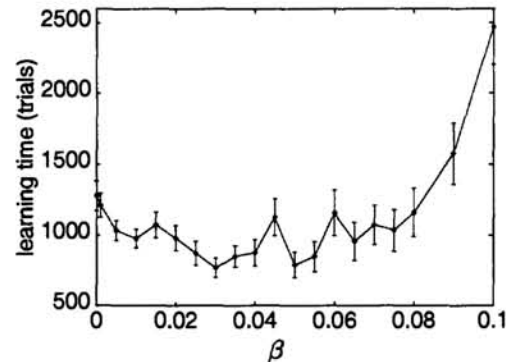

Figure 2: Learning time as a function of the parameter $\beta$ for the bicycle experiment. Each point is an average of 200 runs.

## 3.2  THE CAR ON THE HILL

The second example is Boyan and Moore's mountain-car task (Boyan & Moore, 1995; Singh & Sutton, 1996; Sutton, 1996). Consider driving an under-powered car up a steep mountain road. The problem is that gravity is stronger than the car's engine, and the car cannot accelerate up the slope. The agent must first move the car away from the goal and

up the opposite slope, and then apply full throttle and build up enough momentum to reach the goal. The reward at each time step is $-1$ until the agent reaches the goal, where it receives reward 0. The agent must choose one of three possible actions at each time step: full thrust forward, no thrust, or full thrust backwards. Refer to Singh & Sutton (1996) for the equations of the task.

We used one of the Sarsa-agents with five $9 \times 9$ CMAC tilings that have been thoroughly examined by Singh & Sutton (1996) . The agent's parameters are $\lambda = 0.9$, $\alpha = 0.7$, $\gamma = 1$, and a greedy selection of actions. (These are the best values found by Singh and Sutton.) As in Singh and Sutton's treatment of the problem, all agents were tried for 20 trials, where a trial is one run from a randomly selected starting state to the goal. All the agents used the same set of starting states. The performance measure is the average trial time over the first 20 trials. Figure 3 shows results for two of our simulations. Obviously the action-to-action weights are of no use to the agent, since the lowest point is at $\beta = 0$.

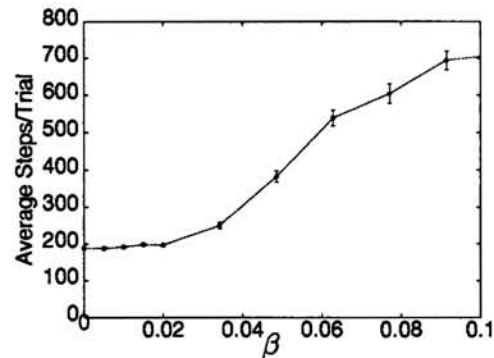

Figure 3: Average trial time of the 20 trials as a function of the parameter $\beta$ for the car on the hill. Each point is an average of 200 runs.

### 3.3 THE RACE TRACK PROBLEM

In the race track problem, which originally was presented by Barto, Bradtke & Singh (1995), the agent controls a car in a race track. The agent must guide the car from the start line to the finish line in the least number of steps possible. The exact position on the start line is randomly selected. The state is given by the position and velocity $(p_x, p_y, v_x, v_y)$ (all integer values). The total number of reachable

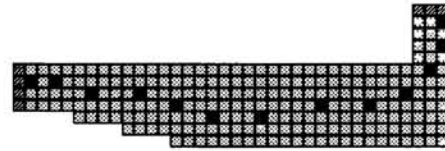

Figure 4: An example of a near-optimal path for the race-track problem. Starting line to the left and finish line at the upper right.

states is 9115 for the track shown in Fig. 4. At each step, the car can accelerate with $a \in \{-1, 0 + 1\}$ in both dimensions. Thus, the agent has 9 possible combinations of actions to choose from. Figure 4 shows positions on a near-optimal path. The agent receives a reward of $-1$ for each step it makes without reaching the goal, and $-2$ for hitting the boundary of the track. Besides the punishment for hitting the boundary of the track, and the fact that the agent's choice of action is always carried out, the problem is as stated in Barto, Bradtke & Singh (1995) and Rummery (1995). The agent's parameters are $\alpha = 0.5$, $\lambda = 0.8$ and $\gamma = 0.98$.

The learning process is divided into epochs consisting of 10 trials each. We consider the task learned if the agent has navigated the car from start to goal in an average of less than 20 time steps for one full epoch. The learning time is defined as the number of the first epoch for which the criterion is met. This learning criterion emphasizes stable learning—the agent needs to be able to solve the problem several times in a row.

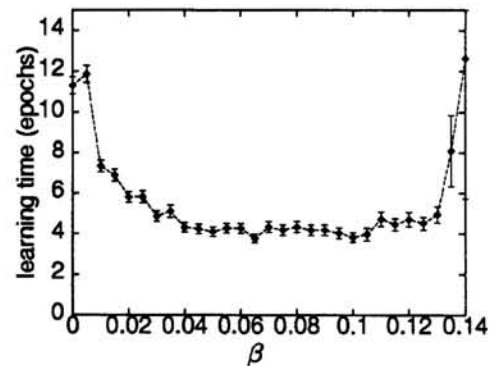

Figure 5: Learning time as a function of the parameter $\beta$ for the race track. Each point is an average of 200 runs.

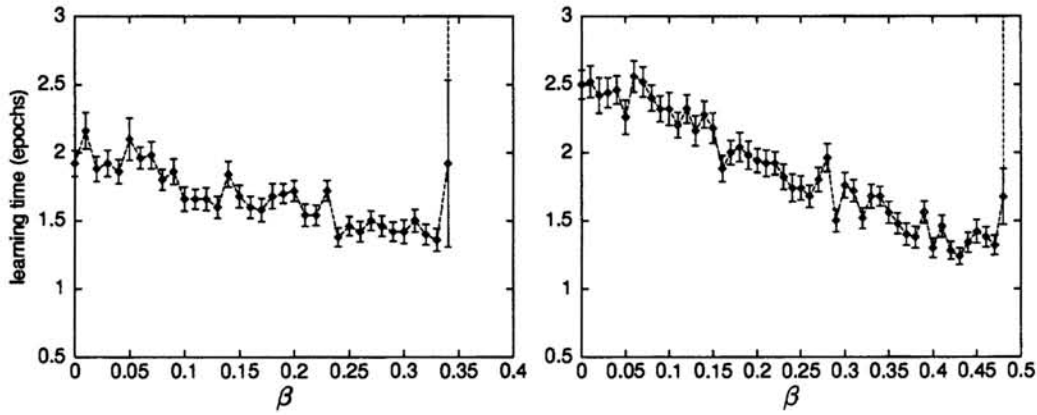

Figure 6: Learning time as a function of the parameter $\beta$ for grid-world tasks: A 3-dimensional grid-world with 216 states (left) and a 4-dimensional grid-world with 256 states (right). All points are averages of 50 runs.

Figure 5 shows how the learning time varies with the value of $\beta$. For a large range of small values of $\beta$ we see a considerable reduction in learning time from 11.5 epochs to 4.2 epochs. As before, large values of $\beta$ slow down learning.

## 3.4 GRID-WORLD TASKS

We tried the new method on a set of grid-world problems in 3, 4 and 5 dimensions. In all the problems the starting point is located at $(1, 1, \ldots)$. For 3 dimensions the goal is located at $(4, 6, 4)$, in 4 dimensions at $(2, 4, 2, 4)$ and in 5 dimensions at $(2, 4, 2, 4, 2)$.

For a $d$-dimensional problem, the agent has $2d$ actions to choose from. Action $2i - 1$ is to move by $-1$ in the $i$th dimension, and action $2i$ is to move by $+1$ in the $i$th dimension. The agent receives a reward of $-0.1$ for each step it makes without reaching the goal, and $+1$ for reaching the goal. If the agent tries to step outside the boundary of the world it maintains its position. The 3-dimensional problem takes place in a $6 \times 6 \times 6$ grid-world, while the 4- and 5-dimensional worlds have each dimension of size 4. Again, the learning process is divided into epochs consisting of 10 trials each. The task is considered learned if the agent has navigated from start to goal in an average of less than some fixed number (15 for 3 dimensions, 19 for 4 and 50 for 5 dimensions) for one full epoch. The agent uses $\alpha = 0.5$, $\lambda = 0.9$ and $\gamma = 0.98$.

Figures 6 and 7 show our results for the grid-world tasks. The learning time is reduced a lot. The usefullness of our new method seems to improve with the number of actions: the more actions the better it works.

Figure 8 shows one of the more clear (but not untypical) set of values for the action-to-action weights for the 3-dimensional

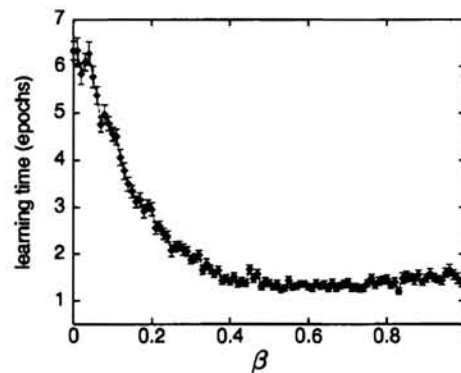

Figure 7: Learning time as a function of the parameter $\beta$ for a 5-dimensional grid-world with 1024 states. All points are averages of 50 runs.

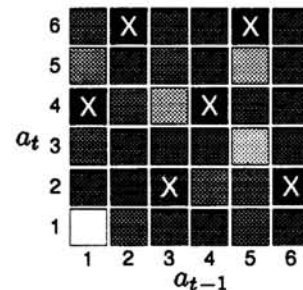

Figure 8: The values of the action-to-action weights; the darker the square the stronger the relationship.

problem. Recommended actions are marked with a white 'X'. The agent has learned two macro-actions. If the agent has performed action number 4 it will continue to perform action 4 all other things being equal. The other macro-action consists of cycling between action 2 and 6. This is a reasonable choice, as one route to the goal consists of performing the actions (44444) and then (262626).

## 3.5   A TASK WITH MANY ACTIONS

Finally we tried a problem with a large number of actions. The world is a 10 times 10 meter square. Instead of picking a dimension to advance in, the agent chooses a direction. The angular space consists of 36 parts of $10°$. The exact position of the agent is discreetized in boxes of 0.1 times 0.1 meter. The goal is a square centered at $(9.5, 7.5)$ with sides measuring 0.4 m. The agent moves 0.3 m per time step, and receives a reward of $+1$ for reaching the goal and $-0.1$ otherwise. The task is considered learned if the agent has navigated from start to goal in an average of less than 200 time steps for one full epoch (10 trials).

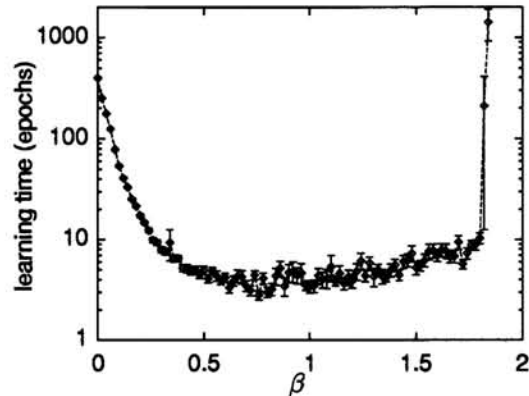

Figure 9: Learning time as a function of the parameter $\beta$. All points are averages of 50 runs. Note the logarithmic scale.

Figure 9 shows the learning curve. The learning time is reduced by a factor of 147 from 397 ($\pm7$) to 2.7 ($\pm0.2$). The only real difference compared to the grid-world problems is the number of actions. The results therefore indicate that the larger the number of actions the better the method works.

## 4   CONCLUSION AND DISCUSSION

We presented a new method for calculating $Q$-values that mix the conventional $Q$-values for the state-to-action mapping with $Q$-values for an action-to-action mapping. We tested the method on a number of problems and found that for all problems except one, the method reduces the total learning time. Furthermore, the agent found macros and learned them. A value function based on values from both state-action and action-action pairs is not guaranteed to converge. Indeed for large values of $\beta$ the method seems unstable, with large variances in the learning time. A good strategy could be to start with a high initial $\beta$ and gradually decrease the value. The empirical results indicate that the usefulness of the method depends on the number of actions: the more actions the better it works. This is also intuitively reasonable, as the information content of the knowledge that a particular action was performed is higher if the agent has more actions to choose from.

### Acknowledgment

The author wishes to thank Andrew G. Barto, Preben Alstrøm, Doina Precup and Amy McGovern for useful comments and suggestions on earlier drafts of this paper and Richard Sutton and Matthew Schlesinger for helpful discussion. Also a lot of thanks to David Cohen for his patience with later than last-minute corrections.

## Footnotes

[1]This is a special case of definitions of macro-actions seen elsewhere. Some researchers take macro-actions to consist of a policy, terminal conditions and an input set (Precup & Sutton, 1998; Sutton, Precup & Singh, 1998; Sutton, Singh, Precup & Ravindran, 1999) while others define it as a local policy (Hauskrecht et al., 1998).

[2]If one action is taken in a state, we allow the traces for the other actions to continue decaying instead of cutting them to 0, contrary to Singh and Sutton (Singh & Sutton, 1996).

# References

Andreae, J. H. & Cashin, P. M. (1969). A learning machine with monologue. *International Journal of Man-Machine Studies, 1*, 1–20.

Barto, A. G., Bradtke, S. J. & Singh, S. (1995). Learning to act using real-time dynamic programming. *Artificial Intelligence, 72*, 81–138.

Boyan, J. A. & Moore, A. W. (1995). Generalization in reinforcement learning: Safely approximating the value function. In *NIPS 7*. (pp. 369–376). The MIT Press.

Burgard, W., Cremers, A. B., Fox, D., Haehnel, D., Lakemeyer, G., Schulz, D., Steiner, W. & Thrun, S. (1998). The interactive museum tour-guide robot. In *Fifteenth National Conference on Artificial Intelligence*.

Gullapalli, V. (1992). *Reinforcement Learning and Its Application to Control.* PhD thesis, University of Massachusetts. COINS Technical Report 92-10.

Hansen, E., Barto, A, & Zilberstein, S. (1997) Reinforcement learning for mixed open-loop and closed-loop control. In *NIPS 9*. The MIT Press.

Hauskrecht, M., Meuleau, N., Boutilier, C., Kaelbling, L. P. & Dean, T. (1998). Hierarchical solution of markov decision processes using macro-actions. In *Proceedings of the Fourteenth International Conference on Uncertainty In Artificial Intelligence*.

Iba, G. A. (1989). A heuristic approach to the discovery of macro-operators. *Machine Learning, 3*.

Korf, R. E. (1985a). Learning to solve problems by searching for macro-operators. *Research Notes in Artificial Intelligence, 5*.

Korf, R. E. (1985b). Macro-operators: A weak method for learning. *Artificial Intelligence, 26*, 35–77.

McCallum, R. A. (1995). *Reinforcement Learning with Selective Perception and Hidden State.* PhD thesis, University of Rochester.

McGovern, A. & Sutton, R. S. (1998). Macro-actions in reinforcement learning: An empirical analysis. Technical Report 98-70, University of Massachusetts.

McGovern, A., Sutton, R. S. & Fagg, A. H. (1997). Roles of macro-actions in accelerating reinforcement learning. In *1997 Grace Hopper Celebration of Women in Computing*.

Precup, D. & Sutton, R. S. (1998). Multi-time models for temporally abstract planning. In *NIPS 10*. The MIT Press.

Randløv, J. & Alstrøm, P. (1998). Learning to drive a bicycle using reinforcement learning and shaping. In *Proceedings of the 15th International Conference on Machine Learning*.

Rummery, G. A. (1995). *Problem Solving with Reinforcement Learning.* PhD thesis, Cambridge University Engineering Department.

Rummery, G. A. & Niranjan, M. (1994). On-line Q-learning using connectionist systems. Technical Report CUED/F-INFENG/TR 166, Engineering Department, Cambridge University.

Singh, S. P. & Sutton, R. S. (1996). Reinforcement learning with replacing eligibility traces. *Machine Learning, 22*, 123–158.

Sutton, R. S. (1996). Generalization in reinforcement learning: Successful examples using sparse coarse coding. In *NIPS 8*. (pp. 1038–1044). The MIT Press.

Sutton, R. S. & Barto, A. G. (1998). *Introduction to Reinforcement Learning.* MIT Press/Bradford Books.

Sutton, R. S., Precup, D. & Singh, S. (1998). Between MDPs and semi-MDPs: Learning, planning, and representing knowledge at multiple temporal scales. Technical Report UM-CS-1998-074, Department of Computer Science, UMass.

Sutton, R. S., Singh, S., Precup, D. & Ravindran, B. (1999). Improved switching among temporally abstract actions. In *NIPS 11*. The MIT Press.
